# A Probabilistic Approach for Optimizing Spectral Clustering

**Rong Jin**[*], **Chris Ding**[†], **Feng Kang**[*]

[*]Lawrence Berkeley National Laboratory, Berkeley, CA 94720
[†]Michigan State University, East Lansing , MI 48824

## Abstract

Spectral clustering enjoys its success in both data clustering and semi-supervised learning. But, most spectral clustering algorithms cannot handle multi-class clustering problems directly. Additional strategies are needed to extend spectral clustering algorithms to multi-class clustering problems. Furthermore, most spectral clustering algorithms employ hard cluster membership, which is likely to be trapped by the local optimum. In this paper, we present a new spectral clustering algorithm, named "Soft Cut". It improves the normalized cut algorithm by introducing soft membership, and can be efficiently computed using a bound optimization algorithm. Our experiments with a variety of datasets have shown the promising performance of the proposed clustering algorithm.

## 1 Introduction

Data clustering has been an active research area with a long history. Well-known clustering methods include the K-means methods (Hartigan & Wong., 1994), Gaussian Mixture Model (Redner & Walker, 1984), Probabilistic Latent Semantic Indexing (PLSI) (Hofmann, 1999), and Latent Dirichlet Allocation (LDA) (Blei et al., 2003). Recently, spectral clustering methods (Shi & Malik, 2000; Ng et al., 2001; Zha et al., 2002; Ding et al., 2001; Bach & Jordan, 2004)have attracted more and more attention given their promising performance in data clustering and simplicity in implementation. They treat the data clustering problem as a graph partitioning problem. In its simplest form, a minimum cut algorithm is used to minimize the weights (or similarities) assigned to the removed edges. To avoid unbalanced clustering results, different objectives have been proposed, including the ratio cut (Hagen & Kahng, 1991), normalized cut (Shi & Malik, 2000) and min-max cut (Ding et al., 2001).

To reduce the computational complexity, most spectral clustering algorithms use the relaxation approach, which maps discrete cluster memberships into continuous real numbers. As a result, it is difficult to directly apply current spectral clustering algorithms to multi-class clustering problems. Various strategies (Shi & Malik, 2000; Ng et al., 2001; Yu & Shi, 2003) have been used to extend spectral clustering algorithms to multi-class clustering problems. One common approach is to first construct a low-dimension space for data representation using the smallest eigenvectors of a graph Laplacian that is constructed based on the pair wise similarity of data. Then, a standard clustering algorithm, such as the K-means method, is applied to cluster data points in the low-dimension space.

One problem with the above approach is how to determine the appropriate number of eigenvectors. A too small number of eigenvectors will lead to an insufficient representation of data, and meanwhile a too large number of eigenvectors will bring in a significant amount of noise to the data representation. Both cases will degrade the quality of clustering. Although it has been shown in (Ng et al., 2001) that the number of required eigenvectors is generally equal to the number of clusters, the analysis is valid only when data points of different clusters are well separated. As will be shown later, when data points are not well separated, the optimal number of eigenvectors can be different from the number of clusters.

Another problem with the existing spectral clustering algorithms is that they are based on binary cluster membership and therefore are unable to express the uncertainty in data clustering. Compared to hard cluster membership, probabilistic membership is advantageous in that it is less likely to be trapped by local minimums. One example is the Bayesian clustering method (Redner & Walker, 1984), which is usually more robust than the K-means method because of its soft cluster memberships. It is also advantageous to use probabilistic memberships when the cluster memberships are the intermediate results and will be used for other processes, for example selective sampling in active learning (Jin & Si, 2004).

In this paper, we present a new spectral clustering algorithm, named "Soft Cut", that explicitly addresses the above two problems. It extends the normalized cut algorithm by introducing probabilistic membership of data points. By encoding membership of multiple clusters into a set of probabilities, the proposed clustering algorithm can be applied directly to multi-class clustering problems. Our empirical studies with a variety of datasets have shown that the soft cut algorithm can substantially outperform the normalized cut algorithm for multi-class clustering.

The rest paper is arranged as follows. Section 2 presents the related work. Section 3 describes the soft cut algorithm. Section 4 discusses the experimental results. Section 5 concludes this study with the future work.

## 2 Related Work

The key idea of spectral clustering is to convert a clustering problem into a graph partitioning problem.

Let $n$ be the number of data points to be clustered. Let $\mathbf{W} = [w_{i,j}]_{n \times n}$ be the weight matrix where each $w_{i,j}$ is the similarity between two data points. For the convenience of discussion, $w_{i,i} = 0$ for all data points. Then, a clustering problem can be formulated into the minimum cut problem, i.e.,

$$\mathbf{q}^* = \arg \min_{\mathbf{q} \in \{-1,1\}^n} \sum_{i,j=1}^{n} w_{i,j}(q_i - q_j)^2 = \mathbf{q}^T \mathbf{L} \mathbf{q} \qquad (1)$$

where $\mathbf{q} = (q_1, q_2, ..., q_n)$ is a vector for binary memberships and each $q_i$ can be either $-1$ or $1$. $\mathbf{L}$ is the Laplacian matrix. It is defined as $\mathbf{L} = \mathbf{D} - \mathbf{W}$, where $\mathbf{D} = [d_{i,i}]_{n \times n}$ is a diagonal matrix with each element $d_{i,i} = \delta_{i,j} \sum_{j=1}^{n} w_{i,j}$. Directly solving the problem in (1) requires combinatorial optimization, which is computationally expensive. Usually, a relaxation approach (Chung, 1997) is used to replace the vector $\mathbf{q} \in \{-1, 1\}^n$ with a vector $\hat{\mathbf{q}} \in \mathbf{R}^n$ under the constraint $\sum_{i=1}^{n} \hat{q}_i^2 = n$. As a result of the relaxation, the approximate solution to (1) is the second smallest eigenvector of Laplacian $L$.

One problem with the minimum cut approach is that it does not take into account the size of clusters, which can lead to clusters of unbalanced sizes. To resolve this problem, several different criteria are proposed, including the ratio cut (Hagen & Kahng, 1991), normalized cut (Shi & Malik, 2000) and min-max cut (Ding et al., 2001). For example, in

the normalized cut algorithm, the following objective is used:

$$J_n(\mathbf{q}) = \frac{C_{+,-}(\mathbf{q})}{D_+(\mathbf{q})} + \frac{C_{+,-}(\mathbf{q})}{D_-(\mathbf{q})} \tag{2}$$

where $C_{+,-}(\mathbf{q}) = \sum_{i,j=1}^n w_{i,j}\delta(q_i,+)\delta(q_j,-)$ and $D_\pm = \sum_{i=1}^n \delta(q_i,\pm)\sum_{j=1}^n w_{i,j}$. In the above objective, the size of clusters, i.e., $D_\pm$, is used as the denominators to avoid clusters of too small size. Similar to the minimum cut approach, a relaxation approach is used to convert the problem in (2) into a eigenvector problem. For multi-class clustering, we can extend the objective in (2) into the following form:

$$J_{norm\_mc}(\mathbf{q}) = \sum_{z=1}^K \sum_{z' \neq z} \frac{C_{z,z'}(\mathbf{q})}{D_z(\mathbf{q})} \tag{3}$$

where $K$ is the number of clusters, vector $\mathbf{q} \in \{1, 2, ..., K\}^n$, $C_{z,z'} = \sum_{i,j=1}^n \delta(q_i,z)\delta(q_j,z')w_{i,j}$, and $D_z = \sum_{i=1}^n \sum_{j=1}^n \delta(q_i,z)w_{i,j}$. However, efficiently finding the solution that minimizes (3) is rather difficult. In particular, a simple relaxation method cannot be applied directly here. In the past, several heuristic approaches (Shi & Malik, 2000; Ng et al., 2001; Yu & Shi, 2003) have been proposed for finding approximate solutions to (3). One common strategy is to first obtain the $K$ smallest (excluding the one with zero eigenvalue) eigenvectors of Laplacian $\mathbf{L}$, and project data points onto the low-dimension space that is spanned by the $K$ eigenvectors. Then, a standard clustering algorithm, such as the K-means method, is applied to cluster data points in this low-dimension space. In contrast to these approaches, the proposed spectral clustering algorithm deals with the multi-class clustering problem directly. It estimates the probabilities for each data point be in different clusters simultaneously. Through the probabilistic cluster memberships, the proposed algorithm will be less likely to be trapped by local minimums, and therefore will be more robust than the existing spectral clustering algorithms.

## 3 Spectral Clustering with Soft Membership

In this section, we describe a new spectral clustering algorithm, named "**Soft Cut**", which extends the normalized cut algorithm by introducing probabilistic cluster membership. In the following, we will present a formal description of the soft cut algorithm, followed by the procedure that efficiently optimizes the related optimization problem.

### 3.1 Algorithm Description

First, notice that $D_z$ in (3) can be expanded as $D_z = \sum_{j=1}^K C_{i,j}$. Thus, the objective function for multi-class clustering in (3) can be rewritten as:

$$J_{n\_mc}(\mathbf{q}) = \sum_{z=1}^K \sum_{z' \neq z} \frac{C_{z,z'}(\mathbf{q})}{D_z(\mathbf{q})} = K - \sum_{z=1}^K \frac{C_{z,z}(\mathbf{q})}{D_z(\mathbf{q})} \tag{4}$$

Let $J'_{n\_mc} = \sum_{z=1}^K \frac{C_{z,z}(\mathbf{q})}{D_z(\mathbf{q})}$. Thus, instead of minimizing $J_{n\_mc}$, we can maximize $J'_{n\_mc}$.

To extend the above objective function to a probabilistic framework, we introduce the probabilistic cluster membership. Let $q_{z,i}$ denote the probability for the $i$-th data point to be in the $z$-th cluster. Let matrix $\mathbf{Q} = [q_{z,i}]_{K \times n}$ include all probabilities $q_{z,i}$. Using the probabilistic notations, we can rewrite $C_{z,z'}$ and $D_z$ as follows:

$$C_{z,z'}(\mathbf{Q}) = \sum_{i,j=1}^n q_{z,i}q_{z',j}w_{i,j}, \quad D_z(\mathbf{Q}) = \sum_{i,j=1}^n q_{z,i}w_{i,j} \tag{5}$$

Substituting the probabilistic expression for $C_{z,z'}$ and $D_z$ into $J'_{n\_mc}$, we have the following optimization problem for probabilistic spectral clustering:

$$\mathbf{Q}^* = \arg\min_{\mathbf{Q}\in\mathbf{R}^{K\times n}} J_{prob}(\mathbf{Q}) = \arg\max_{\mathbf{Q}\in\mathbf{R}^{K\times n}} \sum_{z=1}^{K} \frac{\sum_{i,j=1}^{n} q_{z,i}q_{z,j}w_{i,j}}{\sum_{i,j=1}^{n} q_{z,i}w_{i,j}}$$

$$\text{s.t.}\forall i\in[1..n], z\in[1..K]: q_{z,i}\geq 0, \sum_{z=1}^{K} q_{z,i}=1 \tag{6}$$

### 3.2 Optimization Procedure

In this subsection, we present a bound optimization algorithm (Salakhutdinov & Roweis, 2003) for efficiently finding the solution to (6). It maximizes the objective function in (6) iteratively. In each iteration, a concave lower bound is first constructed for the objective function based on the solution obtained from the previous iteration. Then, a new solution for the current iteration is obtained by maximizing the lower bound. The same procedure is repeated until the solution converges to a local maximum.

Let $\mathbf{Q}' = [q'_{i,j}]_{K\times n}$ be the probabilities obtained in the previous iteration, and $\mathbf{Q} = [q_{i,j}]_{K\times n}$ be the probabilities for current iteration. Define

$$\Delta(\mathbf{Q}, \mathbf{Q}') = \log\frac{J_{prob}(\mathbf{Q})}{J_{prob}(\mathbf{Q}')}$$

which is the logarithm of the ratio of the objective functions between two consecutive iterations. Using the convexity of logarithm function, i.e., $\log(\sum_i p_i q_i) \geq \sum_i p_i \log(q_i)$ for a pdf $\{p_i\}$, we have $\Delta(\mathbf{Q}, \mathbf{Q}')$ lower bound by the following expression:

$$\begin{aligned}
\Delta(\mathbf{Q}, \mathbf{Q}') &= \log\left(\sum_{z=1}^{K}\frac{C_{z,z}(\mathbf{Q})}{D_z(\mathbf{Q})}\right) - \log\left(\sum_{z=1}^{K}\frac{C_{z,z}(\mathbf{Q}')}{D_z(\mathbf{Q}')}\right) \\
&\geq \sum_{z=1}^{K} t_z\left(\log\frac{C_{z,z}(\mathbf{Q})}{C_{z,z}(\mathbf{Q}')} - \log\frac{D_z(\mathbf{Q})}{D_z(\mathbf{Q}')}\right)
\end{aligned} \tag{7}$$

where $t_z$ is defined as:

$$t_z = \frac{\frac{C_{z,z}(\mathbf{Q}')}{D_z(\mathbf{Q}')}}{\sum_{z'=1}^{K}\frac{C_{z',z'}(\mathbf{Q}')}{D_{z'}(\mathbf{Q}')}} \tag{8}$$

Now, the first term within the big bracket in (7), i.e., $\log\frac{C_{z,z}(\mathbf{Q})}{C_{z,z}(\mathbf{Q}')}$, can be further relaxed as:

$$\begin{aligned}
\log\frac{C_{z,z}(\mathbf{Q})}{C_{z,z}(\mathbf{Q}')} &= \log\left(\sum_{i,j=1}^{n}\frac{q'_{z,i}q'_{z,j}w_{i,j}}{C_{z,z}(\mathbf{Q}')}\frac{q_{z,i}q_{z,j}}{q'_{z,i}q'_{z,j}}\right) \\
&\geq 2\sum_{i=1}^{n}\left(\sum_{j=1}^{n} s_z^{i,j}\right)\log(q_{z,i}) - \sum_{i,j=1}^{n} s_z^{i,j}\log(q'_{z,i}q'_{z,j})
\end{aligned} \tag{9}$$

where $s_z^{i,j}$ is defined as:

$$s_z^{i,j} = \frac{q'_{z,i}q'_{z,j}w_{i,j}}{C_{z,z}(\mathbf{Q}')} \tag{10}$$

Meanwhile, using the inequality $\log x \leq x - 1$, we have $\log \frac{D_z(\mathbf{Q})}{D_z(\mathbf{Q}')}$ upper bounded by the following expression:

$$\log \frac{D_z(\mathbf{Q})}{D_z(\mathbf{Q}')} \leq \frac{D_z(\mathbf{Q})}{D_z(\mathbf{Q}')} - 1 = \sum_{i=1}^{n} q_{z,i} \sum_{j=1}^{n} \frac{w_{i,j}}{D_z(\mathbf{Q}')} - 1 \tag{11}$$

Putting together (7), (9), and (11), we have a concave lower bound for the objective function in (6), i.e.,

$$\log J_{prob}(\mathbf{Q}) \geq$$

$$\log J_{prob}(\mathbf{Q}') + \Delta_0(\mathbf{Q}') + 2 \sum_{z=1}^{K} \sum_{i,j=1}^{n} t_z s_z^{i,j} \log q_{z,i} - \sum_{z=1}^{K} \sum_{i,j=1}^{n} \frac{q_{z,i} w_{i,j}}{D_z(\mathbf{Q}')} \tag{12}$$

where $\Delta_0(\mathbf{Q}')$ is defined as:

$$\Delta_0(\mathbf{Q}') = -\sum_{z=1}^{K} t_z \sum_{i,j=1}^{n} s_z^{i,j} w_{i,j} \log(q_{z,i}' q_{z,j}') + 1$$

The optimal solution that maximizes the lower bound in (12) can be computed by setting its derivative to zero, which leads to the following solution:

$$q_{z,i} = \frac{2 t_z \sum_{j=1}^{n} s_z^{i,j}}{t_z \sum_{j=1}^{n} \frac{w_{i,j}}{D_z(\mathbf{Q}')} + \lambda_i} \tag{13}$$

where $\lambda_i$ is a Lagrangian multiplier that ensure $\sum_{z=1}^{K} q_{z,i} = 1$. It can be acquired by maximizing the following objective function:

$$l(\lambda_i) = -\lambda_i + 2 \sum_{z=1}^{K} \left( t_z \sum_{j=1}^{n} s_z^{i,j} \right) \log \left( t_z \sum_{j=1}^{n} \frac{w_{i,j}}{D_z(\mathbf{Q}')} + \lambda_i \right) \tag{14}$$

Since the above objective function is concave, we can apply a standard numerical procedure, such as the Newton's method, to efficiently find the value for $\lambda_i$.

## 4  Experiment

In this section, we focus on examining the effectiveness of the proposed soft cut algorithm for multi-class clustering. In particular, we will address the following two research questions:

1. *How effective is the proposed algorithm for data clustering*? We compare the proposed soft cut algorithm to the normalized cut algorithm with various numbers of eigenvectors.

2. *How robust is the proposed algorithm for data clustering*? We evaluate the robustness of clustering algorithms by examining their variance across multiple trials.

### 4.1  Experiment Design

**Datasets**  In order to extensively examine the effectiveness of the proposed soft cut algorithm, a variety of datasets are used in this experiment. They are:

- *Text documents* that are extracted from the 20 newsgroups to form two five-class datasets, named as "M5" and "L5". Each class contain 100 document and there are totally 500 documents.

Table 1: Datasets Description

| Dataset | Description | #Class | #Instance | #Features |
|---------|-------------|--------|-----------|-----------|
| M5 | Text documents | 5 | 500 | 1000 |
| L5 | Text documents | 5 | 500 | 1000 |
| Pendigit | Pen-based handwritting | 10 | 2000 | 16 |
| Ribosome | Ribosome rDNA sequences | 8 | 1907 | 27617 |

- *Pendigit* that comes from the UCI data repository. It contains 2000 examples that belong to 10 different classes.

- *Ribosomal sequences* that are from RDP project (http://rdp.cme.msu.edu/index.jsp). It contains annotated rRNA sequences of ribosome for 2000 different bacteria that belong to 10 different phylum (e.g., classes). Table 1 provides the detailed information regarding each dataset.

**Evaluation metrics** To evaluate the performance of different clustering algorithms, two different metrics are used:

- *Clustering accuracy*. For the datasets that have no more than five classes, clustering accuracy is used as the evaluation metric. To compute clustering accuracy, each automatically generated cluster is first aligned with a true class. The classification accuracy based on the alignment is then computed, and the clustering accuracy is defined as the maximum classification accuracy among all possible alignments.

- *Normalized mutual information*. For the datasets that have more than five classes, due to the expensive computation involved in finding the optimal alignment, we use the normalized mutual information (Banerjee et al., 2003) as the alternative evaluation metric. If $T_u$ and $T_l$ denote the cluster labels and true class labels assigned to data points, the normalized mutual information "nmi" is defined as

$$\text{nmi} = \frac{2I(T_u, T_l)}{(H(T_u) + H(T_l))}$$

where $I(T_u, T_l)$ stands for the mutual information between clustering labels $T_u$ and true class labels $T_l$. $H(T_u)$ and $H(T_l)$ are the entropy functions for $T_u$ and $T_l$, respectively.

Each experiment was run 10 times with different initialization of parameters. The averaged results together with their variance are used as the final evaluation metric.

**Implementation** We follow the paper (Ng et al., 2001) for implementing the normalized cut algorithm. A cosine similarity is used to measure the affinity between any two data points. Both the EM algorithm and the Kmeans methods are used to cluster the data points that are projected into the low-dimension space spanned by the smallest eigenvectors of a graph Laplacian.

### 4.2 Experiment (I): Effectiveness of The Soft Cut Algorithm

The clustering results of both the soft cut algorithm and the normalized cut algorithm are summarized in Table 2. In addition to the Kmeans algorithm, we also apply the EM clustering algorithm to the normalized cut algorithm. In this experiment, the number of eigenvectors used for the normalized cut algorithms is equal to the number of clusters.

First, comparing to both normalized cut algorithms, we see that the proposed clustering algorithm substantially outperform the normalized cut algorithms for all datasets. Second,

Table 2: Clustering results for different clustering methods. Clustering accuracy is used for dataset "L5" and "M5" as the evaluation metric, and normalized mutual information is used for "Pendigit" and "Ribosome" .

|  | Soft Cut | Normalized Cut (Kmeans) | Normalized Cut (EM) |
|---|---|---|---|
| M5 | $89.2 \pm 1.3$ | $83.2 \pm 8.8$ | $62.4 \pm 5.6$ |
| L5 | $69.2 \pm 2.7$ | $64.2 \pm 4.9$ | $45.1 \pm 4.8$ |
| Pendigit | $56.3 \pm 3.8$ | $46.0 \pm 6.4$ | $52.8 \pm 2.0$ |
| Ribosome | $69.7 \pm 2.9$ | $62.2 \pm 9.1$ | $63.2 \pm 3.8$ |

Table 3: Clustering accuracy for normalized cut with embedding in eigenspace with $K$ eigenvectors. K-means is used.

| #Eigenvector | M5 | L5 | Pendigit | Ribosome |
|---|---|---|---|---|
| $K$ | $\mathbf{83.2 \pm 8.8}$ | $64.1 \pm 4.9$ | $46.0 \pm 6.4$ | $62.2 \pm 9.1$ |
| $K+1$ | $77.6 \pm 8.6$ | $\mathbf{69.6 \pm 6.7}$ | $43.3 \pm 9.1$ | $65.9 \pm 5.8$ |
| $K+2$ | $79.7 \pm 8.5$ | $64.1 \pm 5.7$ | $41.6 \pm 9.3$ | $63.4 \pm 4.8$ |
| $K+3$ | $80.2 \pm 6.6$ | $61.4 \pm 5.8$ | $42.9 \pm 9.6$ | $\mathbf{67.2 \pm 7.6}$ |
| $K+4$ | $74.9 \pm 9.2$ | $59.1 \pm 4.7$ | $47.5 \pm 3.7$ | $60.7 \pm 8.4$ |
| $K+5$ | $70.5 \pm 5.7$ | $66.1 \pm 4.7$ | $39.2 \pm 9.3$ | $63.9 \pm 8.2$ |
| $K+6$ | $75.5 \pm 8.6$ | $61.9 \pm 4.7$ | $43.4 \pm 8.3$ | $63.5 \pm 10.4$ |
| $K+7$ | $75.8 \pm 7.5$ | $59.7 \pm 5.6$ | $46.8 \pm 7.3$ | $56.6 \pm 10.7$ |
| $K+8$ | $73.5 \pm 6.6$ | $61.2 \pm 4.7$ | $\mathbf{49.8 \pm 8.9}$ | $54.3 \pm 7.2$ |

comparing to the normalized cut algorithm using the Kmeans method, we see that the soft cut algorithm has smaller variance in its clustering results. This can be explained by the fact that the Kmeans algorithm uses binary cluster membership and therefore is likely to be trapped by local optimums. As indicated in Table 2, if we replace the Kmeans algoirthm with the EM algorithm in the normalized cut algorithm, the variance in clustering results is generally reduced but at the price of degradation in the performance of clustering. Based on the above observation, we conclude that the soft cut algorithm appears to be effective and robust for multi-class clustering.

## 4.3   Experiment (II): Normalized Cut using Different Numbers of Eigenvectors

One potential reason why the normalized cut algorithm perform worse than the proposed algorithm is that the number of clusters may not be the optimal number of eigenvectors. To examine this issue, we test the normalized cut algorithm with different number of eigenvectors. The Kmeans method is used for clustering the eigenvectors. The results of the normalized cut algorithm using different number of eigenvectors are summarized in Table 3. The best performance is highlighted by the bold fold.

First, we clearly see that the best clustering results may not necessarily happen when the number of eigenvectors is exactly equal to the number of clusters. In fact, for three out of four cases, the best performance is achieved when the number of eigenvectors is larger than the number of clusters. This result indicates that the choice of numbers of eigenvectors can have a significant impact on the performance of clustering. Second, comparing the results in Table 3 to the results in Table 2, we see that the soft cut algorithm is still able to outperform the normalized cut algorithm even with the optimal number of eigenvectors. In general, since spectral clustering is originally designed for binary-class classification, it requires an extra step when it is extended to multi-class clustering problems. Hence, the resulting solutions are usually suboptimal. In contrast, the soft cut algorithm directly

targets on multi-class clustering problems, and thus is able to achieve better performance than the normalized cut algorithm.

# 5   Conclusion

In this paper, we proposed a novel probabilistic algorithm for spectral clustering, called "soft cut" algorithm. It introduces probabilistic membership into the normalized cut algorithm and directly targets on the multi-class clustering problems. Our empirical studies with a number of datasets have shown that the proposed algorithm outperforms the normalized cut algorithm considerably. In the future, we plan to extend this work to other applications such as image segmentation.

# References

Bach, F. R., & Jordan, M. I. (2004). Learning spectral clustering. *Advances in Neural Information Processing Systems 16*.

Banerjee, A., Dhillon, I., Ghosh, J., & Sra, S. (2003). Generative model-based clustering of directional data. *Proceedings of the Ninth ACM SIGKDD International Conference on Knowledge Discovery and Data Mining (KDD-2003)*.

Blei, D. M., Ng, A. Y., & Jordan, M. I. (2003). Latent dirichlet allocation. *J. Mach. Learn. Res.*, *3*, 993–1022.

Chung, F. (1997). *Spectral graph theory*. Amer. Math. Society.

Ding, C., He, X., Zha, H., Gu, M., & Simon, H. (2001). A min-max cut algorithm for graph partitioning and data clustering. *Proc. IEEE Int'l Conf. Data Mining*.

Hagen, L., & Kahng, A. (1991). Fast spectral methods for ratio cut partitioning and clustering. *Proceedings of IEEE International Conference on Computer Aided Design* (pp. 10–13).

Hartigan, J., & Wong., M. (1994). A k-means clustering algorithm. *Appl. Statist.*, *28*, 100–108.

Hofmann, T. (1999). Probabilistic latent semantic indexing. *Proceedings of the 22nd Annual ACM Conference on Research and Development in Information Retrieval* (pp. 50–57). Berkeley, California.

Jin, R., & Si, L. (2004). A bayesian approach toward active learning for collaborative filtering. *Proceedings of the 20th conference on Uncertainty in artificial intelligence* (pp. 278–285). Banff, Canada: AUAI Press.

Ng, A., Jordan, M., & Weiss, Y. (2001). On spectral clustering: Analysis and an algorithm. *Advances in Neural Information Processing Systems 14*.

Redner, R. A., & Walker, H. F. (1984). Mixture densities, maximum likelihood and the em algorithm. *SIAM Review*, *26*, 195–239.

Salakhutdinov, R., & Roweis, S. T. (2003). Adaptive overrelaxed bound optimization methods. *Proceedings of the Twentieth International Conference (ICML 2003)* (pp. 664–671).

Shi, J., & Malik, J. (2000). Normalized cuts and image segmentation. *IEEE Transactions on Pattern Analysis and Machine Intelligence*, *22*, 888–905.

Yu, S. X., & Shi, J. (2003). Multiclass spectral clustering. *Proceedings of Ninth IEEE International Conference on Computer Vision*. Nice, France.

Zha, H., He, X., Ding, C., Gu, M., & Simon, H. (2002). Spectral relaxation for k-means clustering. *Advances in Neural Information Processing Systems 14*.
